# Goal-Based Imitation as Probabilistic Inference over Graphical Models

**Deepak Verma**
Deptt of CSE, Univ. of Washington,
Seattle WA- 98195-2350
deepak@cs.washington.edu

**Rajesh P. N. Rao**
Deptt of CSE, Univ. of Washington,
Seattle WA- 98195-2350
rao@cs.washington.edu

## Abstract

Humans are extremely adept at learning new skills by imitating the actions of others. A progression of imitative abilities has been observed in children, ranging from imitation of simple body movements to goal-based imitation based on inferring intent. In this paper, we show that the problem of goal-based imitation can be formulated as one of inferring goals and selecting actions using a learned probabilistic graphical model of the environment. We first describe algorithms for planning actions to achieve a goal state using probabilistic inference. We then describe how planning can be used to bootstrap the learning of goal-dependent policies by utilizing feedback from the environment. The resulting graphical model is then shown to be powerful enough to allow goal-based imitation. Using a simple maze navigation task, we illustrate how an agent can infer the goals of an observed teacher and imitate the teacher even when the goals are uncertain and the demonstration is incomplete.

## 1 Introduction

One of the most powerful mechanisms of learning in humans is learning by watching. Imitation provides a fast, efficient way of acquiring new skills without the need for extensive and potentially dangerous experimentation. Research over the past decade has shown that even newborns can imitate simple body movements (such as facial actions) [1]. While the neural mechanisms underlying imitation remain unclear, recent research has revealed the existence of "mirror neurons" in the primate brain which fire both when a monkey watches an action or when it performs the same action [2].

The most sophisticated forms of imitation are those that require an ability to infer the underlying goals and intentions of a teacher. In this case, the imitating agent attributes not only visible behaviors to others, but also utilizes the idea that others have internal mental states that underlie, predict, and generate these visible behaviors. For example, infants that are about 18 months old can readily imitate actions on objects, e.g., pulling apart a dumbbell shaped object (Fig. 1a). More interestingly, they can imitate this action even when the adult actor accidentally under- or overshot his target, or the hands slipped several times, leaving the goal-state unachieved (Fig. 1b)[3]. They were thus presumably able to infer the actor's goal, which remained unfulfilled, and imitate not the observed action but the intended one.

In this paper, we propose a model for intent inference and goal-based imitation that utilizes probabilistic inference over graphical models. We first describe how the basic problems of planning an action sequence and learning policies (state to action mappings) can be solved through probabilistic inference. We then illustrate the applicability of the learned graphical model to the problems of goal inference and imitation. Goal inference is achieved by utilizing one's own learned model as a substitute for the teacher's. Imitation is achieved by using one's learned policies to reach an inferred goal state. Examples based on the classic maze navigation domain are provided throughout to help illustrate the behavior of the model. Our results suggest that graphical models provide a powerful platform for modeling and implementing goal-based imitation.

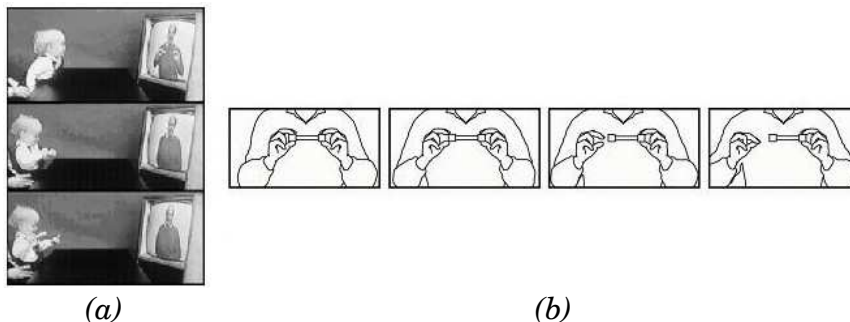

*(a)*                                           *(b)*

Figure 1: **Example of Goal-Based Imitation by Infants:** (a) Infants as young as 14 months old can imitate actions on objects as seen on TV (from [4]). (b) Human actor demonstrating an unsuccessful act. Infants were subsequently able to correctly infer the intent of the actor and successfully complete the act (from [3]).

## 2 Graphical Models

We first describe how graphical models can be used to plan action sequences and learn goal-based policies, which can subsequently be used for goal inference and imitation. Let $\Omega_S$ be the set of states in the environment, $\Omega_A$ the set of all possible actions available to the agent, and $\Omega_G$ the set of possible goals. We assume all three sets are finite. Each goal $g$ represents a target state $Goal_g \in \Omega_S$. At time $t$ the agent is in state $s_t$ and executes action $a_t$. $g_t$ represents the current goal that the agent is trying to reach at time $t$. Executing the action $a_t$ changes the agent's state in a stochastic manner given by the transition probability $P(s_{t+1} \mid s_t, a_t)$, which is assumed to be independent of $t$ i.e., $P(s_{t+1} = s' \mid s_t = s, a_t = a) = \tau_{s'sa}$.

Starting from an initial state $s_1 = s$ and a desired goal state $g$, planning involves computing a series of actions $a_{1:T}$ to reach the goal state, where $T$ represents the maximum number of time steps allowed (the "episode length"). Note that we do not require $T$ to be *exactly* equal to the shortest path to the goal, just as an upper bound on the shortest path length. We use $a, s, g$ to represent a specific value for action, state, and goal respectively. Also, when obvious from the context, we use $s$ for $s_t = s$, $a$ for $a_t = a$ and $g$ for $g_t = g$.

In the case where the state $s_t$ is fully observed, we obtain the graphical model in Fig. 2a, which is also used in *Markov Decision Process* (MDP) [5] (but with a reward function). The agent needs to compute a stochastic *policy* $\hat{\pi}_t(a \mid s, g)$ that maximizes the probability $P(s_{T+1} = Goal_g \mid s_t = s, g_t = g)$. For a large time horizon ($T \gg 1$), the policy is independent of $t$ i.e. $\hat{\pi}_t(a \mid s, g) = \hat{\pi}(a \mid s, g)$ (a *stationary* policy). A more realistic scenario is where the state $s_t$ is hidden but some aspects of it are visible. Given the current state $s_t = s$, an observation $o$ is produced with the probability $P(o_t = o \mid s_t = s) =$

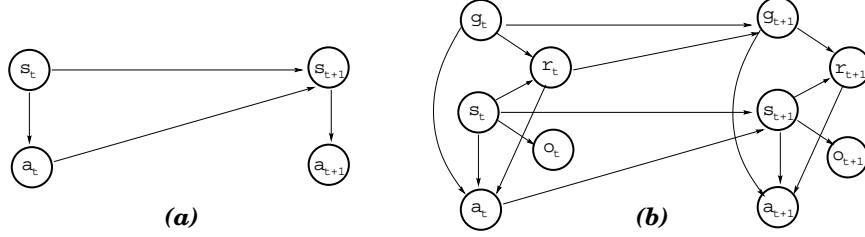

Figure 2: **Graphical Models:** (a) The standard MDP graphical model: The dependencies between the nodes from time step $t$ to $t+1$ are represented by the transition probabilities and the dependency between actions and states is encoded by the policy. (b) The graphical model used in this paper (note the addition of goal, observation and "reached" nodes). See text for more details.

$\zeta_{so}$. In this paper, we assume the observations are discrete and drawn from the set $\Omega_O$, although the approach can be easily generalized to the case of continuous observations (as in HMMs, for example). We additionally include a goal variable $g_t$ and a "reached" variable $r_t$, resulting in the graphical model in Fig. 2b (this model is similar to the one used in partially observable MDPs (POMDPs) but without the goal/reached variables). The goal variable $g_t$ represents the current goal the agent is trying to reach while the variable $r_t$ is a boolean variable that assumes the value 1 whenever the current state equals the current goal state and 0 otherwise. We use $r_t$ to help infer the shortest path to the goal state (given an upper bound $T$ on path length); this is done by constraining the actions that can be selected once the goal state is reached (see next section). Note that $r_t$ can also be used to model the switching of goal states (once a goal is reached) and to implement hierarchical extensions of the present model. The current action $a_t$ now depends not only on the current state but also on the current goal $g_t$, and whether we have reached the goal (as indicated by $r_t$).

**The Maze Domain:** To illustrate the proposed approach, we use the standard stochastic maze domain that has been traditionally used in the MDP and reinforcement learning literature [6, 7]. Figure 3 shows the $7 \times 7$ maze used in the experiments. Solid squares denote a wall. There are five possible actions: `up`, `down`, `left`, `right` and `stayput`. Each action takes the agent into the intended cell with a high probability. This probability is governed by the noise parameter $\eta$, which is the probability that the agent will end up in one of the adjoining (non-wall) squares or remain in the same square. For example, for the maze in Fig. 3, $P([3,5] \mid [4,5], \texttt{left}) = \eta$ while $P([4,4] \mid [4,5], \texttt{left}) = 1 - 3\eta$ (we use `[i,j]` to denote the cell in `i`th row and `j`th column from the top left corner).

## 3 Planning and Learning Policies

### 3.1 Planning using Probabilistic Inference

To simplify the exposition, we first assume full observability ($\zeta_{so} = \delta(s, o)$). We also assume that the environment model $\tau$ is known (the problem of learning $\tau$ is addressed later). The problem of planning can then be stated as follows: Given a goal state $g$, an initial state $s$, and number of time steps $T$, what is the sequence of actions $\hat{a}_{1:T}$ that maximizes the probability of reaching the goal state? We compute these actions using the most probable explanation (MPE) method, a standard routine in graphical model packages (see [7] for an alternate approach). When MPE is applied to the graphical model in Fig. 2b, we obtain:

$$\bar{a}_{1:T}, \bar{s}_{2:T+1}, \bar{g}_{1:T}, \bar{r}_{1:T} = \operatorname{argmax} P(a_{1:T}, s_{2:T}, g_{1:T}, r_{1:T} \mid s_1 = s, s_{T+1} = Goal_g) \quad (1)$$

When using the MPE method, the "reached" variable $r_t$ can be used to compute the shortest path to the goal. For $P(a \mid g, s, r)$, we set the prior for the `stayput` ac-

tion to be very high when $r_t = 1$ and uniform otherwise. This breaks the isomorphism of the MPE action sequences with respect to the `stayput` action, i.e., for $s_1 = [\,4\,,6\,]$, goal$=[\,4\,,7\,]$, and $T = 2$, the probability of `right,stayput` becomes much higher than that of `stayput,right` (otherwise, they have the same posterior probability). Thus, the `stayput` action is discouraged unless the agent has reached the goal. This technique is quite general, in the sense that we can always augment $\Omega_A$ with a *no-op* action and use this technique based on $r_t$ to push the *no-op* actions to the end of a $T$-length action sequence for a pre-chosen upper bound $T$.

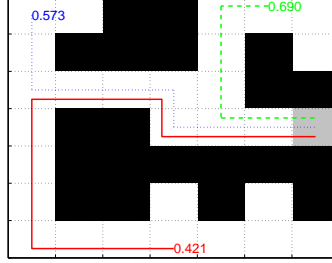

Figure 3: **Planning and Policy Learning:** (a) shows three example plans (action sequences) computed using the MPE method. The plans are shown as colored lines capturing the direction of actions. The numbers denote probability of success of each plan. The longer plans have lower probability of success as expected.

## 3.2 Policy Learning using Planning

Executing a plan in a noisy environment may not always result in the goal state being reached. However, in the instances where a goal state is indeed reached, the executed action sequence can be used to bootstrap the learning of an optimal policy $\hat{\pi}(a \mid s, g)$, which represents the probability for action $a$ in state $s$ when the goal state to be reached is $g$. We define optimality in terms of reaching the goal using the shortest path. Note that the optimal policy may differ from the prior $P(a|s, g)$ which counts all actions executed in state $s$ for goal $g$, regardless of whether the plan was successful.

**MDP Policy Learning:** Algorithm 1 shows a planning-based method for learning policies for an MDP (both $\tau$ and $\pi$ are assumed unknown and initialized to a prior distribution, e.g., uniform). The agent selects a random start state and a goal state (according to $P(g_1)$), infers the MPE plan $\bar{a}_{1:T}$ using the current $\tau$, executes it, and updates the frequency counts for $\tau_{s'sa}$ based on the observed $s_t$ and $s_{t+1}$ for each $a_t$. The policy $\hat{\pi}(a \mid s, g)$ is only updated (by updating the action frequencies) if the goal $g$ was reached. To learn an accurate $\tau$, the algorithm is biased towards exploration of the state space initially based on the parameter $\alpha$ (the "exploration probability"). $\alpha$ decreases by a decay factor $\gamma$ ($0 < \gamma < 1$) with each iteration so that the algorithm transitions to an "exploitation" phase when transition model is well learned and favors the execution of the MPE plan.

**POMDP Policy Learning:** In the case of partial observability, Algorithm 1 is modified to compute the plan $\bar{a}_{1:T}$ based on observation $o_1 = o$ as evidence instead of $s_1 = s$ in Eq.1. The plan is executed to record observations $o_{2:T+1}$, which are then used to compute the MPE estimate for the hidden states:$\bar{s}^o_{1:T+1}, \bar{g}_{1:T}, \bar{r}_{1:T+1} = $ argmax $P(s_{1:T+1}, g_{1:T}, r_{1:T+1} \mid o_{1:T+1}, \bar{a}_{1:T}, g_{T+1} = g)$. The MPE estimate $\bar{s}^o_{1:T+1}$ is then used instead of $s^o_{1:T+1}$ to update $\hat{\pi}$ and $\tau$.

**Results:** Figure 4a shows the error in the learned transition model and policy as a function of the number of iterations of the algorithm. Error in $\tau_{s'sa}$ was defined as the squared sum of differences between the learned and true transition parameters. Error in the learned policy was defined as the number of disagreements between the optimal deterministic pol-

**Algorithm 1** Policy learning in an unknown environment

1: Initialize transition model $\tau_{s'sa}$, policy $\hat{\pi}(a \mid s, g)$, $\alpha$, and $numTrials$.
2: **for** $iter = 1$ to $numTrials$ **do**
3:     Choose random start location $s_1$ based on prior $P(s_1)$.
4:     Pick a goal $g$ according to prior $P(g_1)$.
5:     With probability $\alpha$:
6:         $a_{1:T}=$ Random action sequence.
7:     Otherwise:
8:         Compute MPE plan as in Eq.1 using current $\tau_{s'sa}$.
            Set $a_{1:T} = \bar{a}_{1:T}$
9:     Execute $a_{1:T}$ and record observed states $s^o_{2:T+1}$.
10:     Update $\tau_{s'sa}$ based on $a_{1:T}$ and $s^o_{1:T+1}$.
11:     If the plan was successful, update policy $\hat{\pi}(a \mid s, g)$ using $a_{1:T}$ and $s^o_{1:T+1}$.
12:     $\alpha = \alpha \times \gamma$
13: **end for**

icy for each goal computed via policy iteration and $\operatorname*{argmax}_a \hat{\pi}(a \mid s, g)$, summed over all goals. Both errors decrease to zero with increasing number of iterations. The policy error decreases only after the transition model error becomes significantly small because without an accurate estimate of $\tau$, the MPE plan is typically incorrect and the agent rarely reaches the goal state, resulting in little or no learning of the policy. Figs. 4b shows the maximum probability action $\operatorname*{argmax}_a \hat{\pi}(a \mid s, g)$ learned for each state (maze location) for one of the goals. It is clear that the optimal action has been learned by the algorithm for all locations to reach the given goal state. The results for the POMDP case are shown in Fig. 4c and d. The policy error decreases but does not reach zero because of perceptual ambiguity at certain locations such as corners, where two (or more) actions may have roughly equal probability (see Fig. 4d). The optimal strategy in these ambiguous states is to sample from these actions.

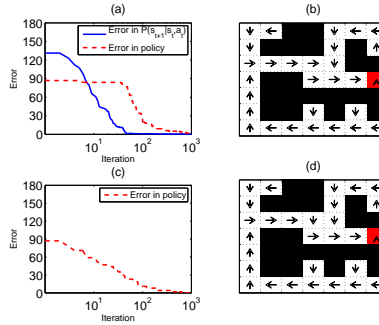

Figure 4: **Learning Policies for an MDP and a POMDP:** (a) shows the error in the transition model and policy w.r.t the true transition model and optimal policy for the maze MDP. (b) The optimal policy learned for one of the 3 goals. (c) and (d) show corresponding results for the POMDP case (the transition model was assumed to be known). The long arrows represent the maximum probability action while the short arrows show all the high probability actions when there is no clear winner.

## 4 Inferring Intent and Goal-Based Imitation

Consider a task where the agent gets observations $o_{1:t}$ from observing a teacher and seeks to imitate the teacher. We use $P(o_t = o \mid s_t = s) = \zeta_{so}$ in Fig. 2b (for the examples here, $\zeta_{so}$ was the same as in the previous section). Also, for $P(a|s, g, r_t = 0)$, we use the policy $\hat{\pi}(a \mid s, g)$ learned as in the previous section. The goal of the agent is to infer the intention

of the teacher given a (possibly incomplete) demonstration and to reach the intended goal using its policy (which could be different from the teacher's optimal policy). Using the graphical model formulation the problem of goal inference reduces to finding the marginal $P(g_T \mid o_{1:t'})$, which can be efficiently computed using standard techniques such as belief propagation. Imitation is accomplished by choosing the goal with the highest probability and executing actions to reach that goal.

Fig. 5a shows the results of goal inference for the set of noisy teacher observations in Fig. 5b. The three goal locations are indicated by red, blue, and green squares respectively. Note that the inferred goal probabilities correctly reflect the putative goal(s) of the teacher at each point in the teacher trajectory. In addition, even though the teacher demonstration is incomplete, the imitator can perform goal-based imitation by inferring the teacher's most likely goal as shown in Fig. 5c. This mimics the results reported by [3] on the intent inference by infants.

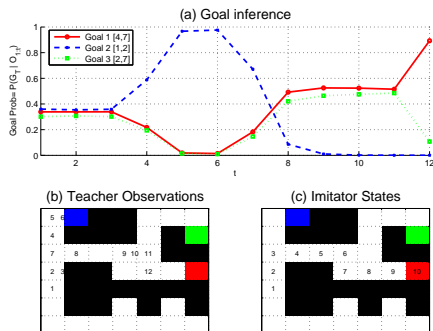

Figure 5: **Goal Inference and Goal-Based Imitation:** (a) shows the goal probabilities inferred at each time step from teacher observations. (b) shows the teacher observations, which are noisy and include a detour while en route to the red goal. The teacher demonstration is incomplete and stops short of the red goal. (c) The imitator infers the most likely goal using (a) and performs goal-based imitation while avoiding the detour (The numbers $t$ in a cell in (b) and (c) represent $o_t$ and $s_t$ respectively).

## 5 Online Imitation with Uncertain Goals

Now consider a task where the goal is to imitate a teacher online (i.e., simultaneously with the teacher). The teacher observations are assumed to be corrupted by noise and may include significant periods of occlusion where no data is available. The graphical model framework provides an elegant solution to the problem of planning and selecting actions when observations are missing and only a probability distribution over goals is available. The best current action can be picked using the marginal $P(a_t \mid o_{1:t})$, which can be computed efficiently for the graphical model in Fig. 2c. This marginal is equal to $\sum_i P(a_t|g_i, o_{1:t})P(g_i|o_{1:t})$, i.e., the policy for each goal *weighted* by the likelihood of that goal given past teacher observations, which corresponds to our intuition of how actions should be picked when goals are uncertain.

Fig. 6a shows the inferred distribution over goal states as the teacher follows a trajectory given by the noisy observations in Fig. 6b. Initially, all goals are nearly equally likely (with a slight bias for the nearest goal). Although the goal is uncertain and certain portions of the teacher trajectory are occluded[1], the agent is still able to make progress towards regions

most likely to contain any probable goal states and is able to "catch-up" with the teacher when observations become available again (Fig.. 6c).

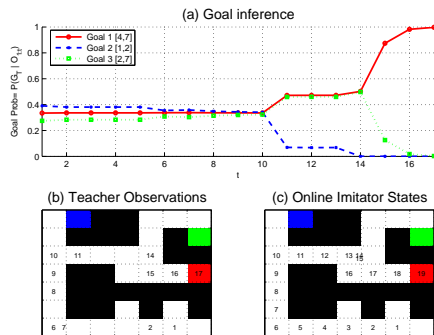

Figure 6: **Online Imitation with Uncertain Goals:** (a) shows the goal probabilities inferred by the agent at each time step for the noisy teacher trajectory in (b). (b) Observations of the teacher. Missing numbers indicate times at which the teacher was occluded. (c) The agent is able to follow the teacher trajectory even when the teacher is occluded based on the evolving goal distribution in (a).

# 6   Conclusions

We have proposed a new model for intent inference and goal-based imitation based on probabilistic inference in graphical models. The model assumes an initial learning phase where the agent explores the environment (cf. body babbling in infants [8]) and learned a graphical model capturing the sensory consequences of motor actions. The learned model is then used for planning action sequences to goal states and for learning policies. The resulting graphical model then serves as a platform for intent inference and goal-based imitation.

Our model builds on the proposals of several previous researchers. It extends the approach of [7] from planning in a traditional state-action Markov model to a full-fledged graphical model involving states, actions, and goals with edges for capturing conditional distributions denoting policies. The indicator variable $r_t$ used in our approach is similar to the ones used in some hierarchical graphical models [9, 10, 11]. However, these papers do not address the issue of action selection or imitation. Several models of imitation have previously been proposed [12, 13, 14, 15, 16, 17]; these models are typically not probabilistic and have focused on trajectory following rather than intent inference and goal-based imitation.

An important issue yet to be resolved is the scalability of the proposed approach. The Bayesian model requires both a learned environment model as well as a learned policy. In the case of the maze example, these were learned using a relatively small number of trials due to small size of the state space. A more realistic scenario involving, for example, a human or a humanoid robot would presumably require an extremely large number of trials during learning due to the large number of degrees-of-freedom available; fortunately, the problem may be alleviated in two ways: first, only a small portion of the state space may be physically realizable due to constraints imposed by the body or environment; second, the agent could selectively refine its models during imitative sessions. Hierarchical state space models may also help in this regard.

The probabilistic model we have proposed also opens up the possibility of applying Bayesian methodologies such as manipulation of prior probabilities of task alternatives to obtain a deeper understanding of goal inference and imitation in humans. For example, one

could explore the effects of biasing a human subject towards particular classes of actions (e.g., through repetition) under particular sets of conditions. One could also manipulate the learned environment model used by subjects with the help of virtual reality environments. Such manipulations have yielded valuable information regarding the type of priors and internal models that the adult human brain uses in perception (see, e.g., [18]) and in motor learning [19]. We believe that the application of Bayesian techniques to imitation could shed new light on the problem of how infants acquire internal models of the people and objects they encounter in the world.

## Footnotes

[1]We simulated occlusion using a special observation symbol which carried no information about current state, i.e., $P(\text{occluded} \mid s) = \epsilon$ for all $s$ ($\epsilon \ll 1$)

## References

[1] A. N. Meltzoff and M. K. Moore. Newborn infants imitate adult facial gestures. *Child Development*, 54:702–709, 1983.

[2] L. Fogassi G. Rizzolatti, L. Fadiga and V. Gallese. From mirror neurons to imitation, facts, and speculations. *In A. N. Meltzoff and W. Prinz (Eds.), The imitative mind: Development, evolution, and brain bases*, pages 247–266, 2002.

[3] A. N. Meltzoff. Understanding the intentions of others: Re-enactment of intended acts by 18-month-old children. *Developmental Psychology*, 31:838–850, 1995.

[4] A. N. Meltzoff. Imitation of televised models by infants. *Child Development*, 59:1221–1229, 1988a.

[5] C. Boutilier, T. Dean, and S. Hanks. Decision-theoretic planning: Structural assumptions and computational leverage. *Journal of AI Research*, 11:1–94, 1999.

[6] R. S. Sutton and A. Barto. *Reinforcement Learning: An Introduction*. MIT Press, Cambridge, MA, 1998.

[7] H. Attias. Planning by probabilistic inference. In *Proceedings of the 9th Int. Workshop on AI and Statistics*, 2003.

[8] R. P. N. Rao, A. P. Shon, and A. N. Meltzoff. A Bayesian model of imitation in infants and robots. In *Imitation and Social Learning in Robots, Humans, and Animals*. Cambridge University Press, 2004.

[9] G. Theocharous, K. Murphy, and L. P. Kaelbling. Representing hierarchical POMDPs as DBNs for multi-scale robot localization. *ICRA*, 2004.

[10] S. Fine, Y. Singer, and N. Tishby. The hierarchical hidden Markov model: Analysis and applications. *Mach. Learn.*, 32(1):41–62, 1998.

[11] H. Bui, D. Phung, and S. Venkatesh. Hierarchical hidden Markov models with general state hierarchy. In *AAAI 2004*, 2004.

[12] G. Hayes and J. Demiris. A robot controller using learning by imitation. *Proceedings of the 2nd International Symposium on Intelligent Robotic Systems, Grenoble, France,*, pages 198–204, 1994.

[13] M. J. Mataric and M. Pomplun. Fixation behavior in observation and imitation of human movement. *Cognitive Brain Research*, 7:191–202, 1998.

[14] S. Schaal. Is imitation learning the route to humanoid robots? *Trends in Cognitive Sciences*, 3:233–242, 1999.

[15] A. Billard and K. Dautenhahn. Experiments in social robotics- grounding and use of communication in robotic agents. *Adaptive Behavior*, 7:3–4, 2000.

[16] C. Breazeal and B. Scassellati. Challenges in building robots that imitate people. *In K. Dautenhahn and C. L. Nehaniv (Eds.), Imitation in animals and artifacts*, pages 363–390, 2002.

[17] K. Dautenhahn and C. Nehaniv. *Imitation in Animals and Artifacts.* Cambridge, MA: MIT Press, 2002.

[18] B. A. Olshausen R. P. N. Rao and M. S. Lewicki (Eds.). *Probabilistic Models of the Brain: Perception and Neural Function.* Cambridge, MA: MIT Press, 2002.

[19] KP. Krding and D. Wolpert. Bayesian integration in sensorimotor learning. *Nature*, 427:244–247, 2004.
